# LINKS BETWEEN MARKOV MODELS AND MULTILAYER PERCEPTRONS

H. Bourlard [†,‡] & C.J. Wellekens [†]

[†]Philips Research Laboratory
Brussels, B-1170 Belgium.

[‡]Int. Comp. Science Institute
Berkeley, CA 94704 USA.

## ABSTRACT

Hidden Markov models are widely used for automatic speech recognition. They inherently incorporate the sequential character of the speech signal and are statistically trained. However, the a-priori choice of the model topology limits their flexibility. Another drawback of these models is their weak discriminating power. Multilayer perceptrons are now promising tools in the connectionist approach for classification problems and have already been successfully tested on speech recognition problems. However, the sequential nature of the speech signal remains difficult to handle in that kind of machine. In this paper, a discriminant hidden Markov model is defined and it is shown how a particular multilayer perceptron with contextual and extra feedback input units can be considered as a general form of such Markov models.

## INTRODUCTION

*Hidden Markov models (HMM)* [Jelinek, 1976; Bourlard et al., 1985] are widely used for automatic isolated and connected speech recognition. Their main advantages lie in the ability to take account of the time sequential order and variability of speech signals. However, the a-priori choice of a model topology (number of states, probability distributions and transition rules) limits the flexibility of the *HMM*'s, in particular speech contextual information is difficult to incorporate. Another drawback of these models is their weak discriminating power. This fact is clearly illustrated in [Bourlard & Wellekens, 1989; Waibel et al., 1988] and several solutions have recently been proposed in [Bahl et al., 1986; Bourlard & Wellekens, 1989; Brown, 1987].

The *multilayer perceptron (MLP)* is now a familiar and promising tool in connectionist approach for classification problems [Rumelhart et al., 1986; Lippmann, 1987] and has already been widely tested on speech recognition problems [Waibel et al., 1988; Watrous & Shastri, 1987; Bourlard & Wellekens, 1989]. However, the sequential nature of the speech signal remains difficult to handle with *MLP*. It is shown here how an *MLP* with contextual and extra feedback input units can be considered as a form of discriminant *HMM*.

## STOCHASTIC MODELS

### TRAINING CRITERIA

Stochastic speech recognition is based on the comparison of an utterance to be recognized with a set of probabilistic finite state machines known as *HMM*. These are trained such that the probability $P(W_i|X)$ that model $W_i$ has produced the associated utterance $X$ must be maximized, but the parameter space which this optimization is performed over makes the difference between independently trained models and *discriminant* ones.

Indeed, the probability $P(W_i|X)$ can be written as

$$P(W_i|X) = \frac{P(X|W_i).P(W_i)}{P(X)} . \tag{1}$$

In a recognition phase, $P(X)$ may be considered as a constant since the model parameters are fixed but, in a training phase, this probability depends on the parameters of all possible models. Taking account of the fact that the models are mutually exclusive and if $\lambda$ represents the parameter set (for all possible models), (1) may then be rewritten as:

$$P(W_i|X,\lambda) = \frac{P(X|W_i,\lambda)P(W_i)}{P(X|W_i,\lambda)P(W_i) + \sum_{k \neq i} P(X|W_k,\lambda)P(W_k)}, \tag{2}$$

Maximization of $P(W_i|X,\lambda)$ as given by (2) is usually simplified by restricting it to the subspace of the $W_i$ parameters. This restriction leads to the *Maximum Likelihood Estimators (MLE)*. The summation term in the denominator is constant over the parameter space of $W_i$ and thus, maximization of $P(X|W_i,\lambda)$ implies that of its bilinear map (2). A language model provides the value of $P(W_i)$ independently of the acoustic decoding [Jelinek, 1976].

On the other hand, maximization of $P(W_i|X,\lambda)$ with respect to the whole parameter space (i.e. the parameters of *all* models $W_1, W_2, \ldots$) leads to discriminant models since it implies that the contribution of $P(X|W_i,\lambda)P(W_i)$ should be enhanced while that of the rival models, represented by

$$\sum_{k \neq i} P(X|W_k,\lambda)P(W_k),$$

should be reduced. This maximization with respect to the whole parameter space has been shown equivalent to the maximization of *Mutual Information (MMI)* between a model and a vector sequence [Bahl et al., 1986; Brown, 1987].

### STANDARD HIDDEN MARKOV MODELS

In the regular discrete *HMM*, the acoustic vectors (e.g. corresponding to 10 ms speech frames) are generally quantized in a front-end processor where each one is replaced by the closest (e.g. according to an Euclidean norm) prototype vector

$y_i$ selected in a predetermined finite set $\mathcal{Y}$ of cardinality $I$. Let $Q$ be a set of $K$ different states $q(k)$, with $k = 1, \ldots, K$. Markov models are then constituted by the association (according to a predefined topology) of some of these states. If *HMM* are trained along the *MLE* criterion, the parameters of the models (defined hereunder) must be optimized for maximizing $P(X|W)$ where $X$ is a training sequence of quantized acoustic vectors $x_n \in \mathcal{Y}$, with $n = 1, \ldots, N$ and $W$ is its associated Markov model made up of $L$ states $q_\ell \in Q$ with $\ell = 1, \ldots, L$. Of course, the same state may occur several times with different indices $\ell$, so that $L \neq K$. Let us denote by $q_\ell^n$ the presence on state $q_\ell$ at a given time $n \in [1, N]$. Since events $q_\ell^n$ are mutually exclusive, probability $P(X|W)$ can be written for any arbitrary $n$:

$$P(X|W) = \sum_{\ell=1}^{L} P(q_\ell^n, X|W), \tag{3}$$

where $P(q_\ell^n, X|W)$ denotes thus the probability that $X$ is produced by $W$ while associating $x_n$ with state $q_\ell$. Maximization of (3) can be worked out by the classical forward-backward recurrences of the *Baum-Welch* algorithm [Jelinek 1976, Bourlard et al., 1985]

Maximization of $P(X|W)$ is also usually approximated by the *Viterbi* criterion. It can be viewed as a simplified version of the *MLE* criterion where, instead of taking account of all possible state sequences in $W$ capable of producing $X$, one merely considers the most probable one. To make all possible paths apparent, (3) can also be rewritten as

$$P(X|W) = \sum_{\ell_1=1}^{L} \cdots \sum_{\ell_N=1}^{L} P(q_{\ell_1}^1, \ldots, q_{\ell_N}^N, X|W),$$

and the explicit formulation of the *Viterbi* criterion is obtained by replacing all summations by a "*max*" operator. Probability (3) is then approximated by:

$$\overline{P}(X|W) = \max_{\ell_1,\ldots,\ell_N} P(q_{\ell_1}^1, \ldots, q_{\ell_N}^N, X|W), \tag{4}$$

and can be calculated by the classical *dynamic time warping (DTW)* algorithm [Bourlard et al., 1985]. In that case, each training vector is then uniquely associated with only one particular transition.

In both cases (*MLE* and *Viterbi*), it can be shown that, according to classical hypotheses, $P(X|W)$ and $\overline{P}(X|W)$ are estimated from the set of local parameters $p[q(\ell), y_i | q^{(-)}(k), W]$, for $i = 1, \ldots, I$ and $k, \ell = 1, \ldots, K$. Notations $q^{(-)}(k)$ and $q(\ell)$ denote states $\in Q$ observed at two consecutive instants. In the particular case of the Viterbi criterion, these parameters are estimated by:

$$\hat{p}[q(\ell), y_i | q^{(-)}(k)] = \frac{n_{ik\ell}}{\sum_{j=1}^{I} \sum_{m=1}^{K} n_{jkm}}, \quad \forall i \in [1, I], \quad \forall k, \ell \in [1, K], \tag{5}$$

where $n_{ik\ell}$ denotes the number of times each prototype vector $y_i$ has been associated with a particular transition from $q(k)$ to $q(\ell)$ during the training. However, if

the models are trained along this formulation of the *Viterbi* algorithm, no discrimination is taken into account. For instance, it is interesting to observe that the local probability (5) is not the suitable measure for the labeling of a prototype vector $y_i$, i.e. to find the most probable state given a current input vector and a specified previous state. Indeed, the decision should ideally be based on the Bayes rule. In that cae, the most probable state $q(\ell_{opt})$ is defined by

$$\ell_{opt} = \operatorname*{argmax}_{\ell} \; p[q(\ell)|y_i, q^{(-)}(k)] \,, \tag{6}$$

and not on the basis of (5).

It can easily be proved that the estimate of the Bayes probabilities in (6) are:

$$\hat{p}[q(\ell)|y_i, q^{(-)}(k)] = \frac{n_{ik\ell}}{\sum_{m=1}^{K} n_{ikm}} \,. \tag{7}$$

In the last section, it is shown that these values can be generated at the output of a particular *MLP*.

## DISCRIMINANT HMM

For quantized acoustic vectors and *Viterbi* criterion, an alternative *HMM* using discriminant local probabilities can also be described. Indeed, as the correct criterion should be based on (1), comparing with (4), the "Viterbi formulation" of this probability is

$$\overline{P}(W|X) = \max_{\ell_1,\ldots,\ell_N} P(q^1_{\ell_1},\ldots,q^N_{\ell_N},W|X) \,. \tag{8}$$

Expression (8) clearly puts the best path into evidence. The right hand side factorizes into

$$P(q^1_{\ell_1},\ldots,q^N_{\ell_N},W|X) = P(q^1_{\ell_1},\ldots,q^N_{\ell_N}|X).P(W|X,q^1_{\ell_1},\ldots,q^N_{\ell_N}) \,.$$

and suggests two separate steps for the recognition. The first factor represents the acoustic decoding in which the acoustic vector sequence is converted into a sequence of states. Then, the second factor represents a phonological and lexical step: once the sequence of states is known, the model $W$ associated with $X$ can be found from the state sequence without an explicit dependence on $X$ so that

$$P(W|X,q^1_{\ell_1},\ldots,q^N_{\ell_N}) = P(W|q^1_{\ell_1},\ldots,q^N_{\ell_N}) \,.$$

For example, if the states represent phonemes, this probability must be estimated from phonological knowledge of the vocabulary once for all in a separate process without any reference to the input vector sequence.

On the contrary, $P(q^1_{\ell_1},\ldots,q^N_{\ell_N}|X)$ is immediately related to the discriminant local probabilities and may be factorized in

$$P(q^1_{\ell_1},\ldots,q^N_{\ell_N}|X) = p(q^1_{\ell_1}|X).p(q^2_{\ell_2}|X,q^1_{\ell_1})\ldots p(q^N_{\ell_N}|X,q^1_{\ell_1},\ldots,q^{N-1}_{\ell_{N-1}}). \tag{9}$$

Now, each factor of (9) may be simplified by relaxing the conditional constraints. More specifically, the factors of (9) are assumed dependent on the previous state only and on a signal window of length $2p + 1$ centered around the current acoustic vector. The current expression of these local contributions becomes

$$p(q^i_{\ell_i}|X, q^1_{\ell_1}, \ldots, q^{i-1}_{\ell_{i-1}}) = p(q^i_{\ell_i}|X^{i+p}_{i-p}, q^{i-1}_{\ell_{i-1}}), \qquad (10)$$

where *input contextual information* is now taken into account, $X^n_m$ denoting the vector sequence $x_m, x_{m+1}, \ldots, x_n$. If input contextual information is neglected ($p = 0$), equation (10) represents nothing else but the discriminant local probability (7) and is at the root of a *discriminant discrete HMM*. Of course, as for (7), these local probabilities could also be simply estimated by counting on the training set, but the exponential increase of the number of parameters with the width $2p + 1$ of the contextual window would require an exceedingly large storage capacity as an excessive size of training data to obtain statistically significant parameters. It is shown in the following section how this drawback is circumvented by using an *MLP*. It is indeed proved that, for the training vectors, the optimal outputs of a *recurrent* and *context-sensitive MLP* are the estimates of the local probabilities (10). Given its so-called "generalization property", the *MLP* can then be used for interpolating on the test set.

Of course, from the local contributions (10), $\overline{P}(W|X)$ can still be obtained by the classical one-stage dynamic programming [Ney, 1984; Bourlard et al., 1985]. Indeed, inside *HMM*, the following dynamic programming recurrence holds

$$\overline{P}(q_\ell|X^n_1) = \max_k \left( \overline{P}(q_k|X^{n-1}_1) . p(q_\ell|X^{n+p}_{n-p}, q_k) \right), \qquad (11)$$

where parameter $k$ runs over all possible states preceding $q_\ell$ and $\overline{P}(q_\ell|X^n_1)$ denotes the cumulated best path probability of reaching state $q_\ell$ and having emitted the partial sequence $X^n_1$ .

## RECURRENT MLP AND DISCRIMINANT HMM

Let $q(k)$, with $k = 1, \ldots, K$, be the output units of an *MLP* associated with different classes (each of them corresponding a particular state of $\mathcal{Q}$) and $I$ the number of prototype vectors $y_i$. Let $v_i$ denote a particular binary input of the *MLP*. If no contextual information is used, $v_i$ is the binary representation of the index $i$ of prototype vector $y_i$ and, more precisely, a vector with all zero components but the $i$-th one equal to 1. In the case of contextual input, vector $v_i$ is obtained by concatenating several representations of prototype vectors belonging to a given contextual window centered on a current $y_i$. The architecture of the resulting *MLP* is then similar to NETtalk initially described in [Sejnowski & Rosenberg, 1987] for mapping written texts to phoneme strings. The same kind of architecture has also been proved successful in performing the classification of acoustic vector strings into phoneme strings, where each current vector was classified by taking account

of its surrounding vectors [Bourlard & Wellekens, 1989]. The input field is then constituted by several groups of units, each group representing a prototype vector. Thus, if $2p+1$ is the width of the contextual window, there are $2p+1$ groups of $I$ units in the input layer.

However, since each acoustic vector is classified independently of the preceding classifications in such feedforward architectures, the sequential character of the speech signal is not modeled. The system has no short-term memory from one classification to the next one and successive classifications may be contradictory. This phenomenon does not appear in *HMM* since only some state sequences are permitted by the particular topology of the model.

Let us assume that the training is performed on a sequence of $N$ binary inputs $\{v_{i_1}, \ldots, v_{i_N}\}$ where each $i_n$ represents the index of the prototype vector at time $n$ (if no context) or the "index" of one of the $I^{(2p+1)}$ possible inputs (in the case of a $2p+1$ contextual window). Sequential classification must rely on the previous decisions but the final goal remains the association of the current input vectors with their own classes. An *MLP* achieving this task will generate, for each current input vector $v_{i_n}$ and each class $q(\ell)$, $\ell = 1, \ldots, K$, an output value $g(i_n, k_n, \ell)$ depending on the class $q(k_n)$ in which the preceding input vector $v_{i_{n-1}}$ was classified. Supervision comes from the a-priori knowledge of the classification of each $v_{i_n}$. The training of the *MLP* parameters is usually based on the minimization of a *mean square criterion (LMSE)* [Rumelhart et al., 1986] which, with our requirements, takes the form:

$$E = \frac{1}{2} \sum_{n=1}^{N} \sum_{\ell=1}^{K} \left[ g(i_n, k_n, \ell) - d(i_n, \ell) \right]^2 , \tag{12}$$

where $d(i_n, \ell)$ represents the target value of the $\ell$-th output associated with the input vector $v_{i_n}$. Since the purpose is to associate each input vector with a single class, the target outputs, for a vector $v_i \in q(\ell)$, are:

$$d(i, \ell) = 1,$$
$$d(i, m) = 0, \quad \forall m \neq \ell ,$$

which can also be expressed, for each particular $v_i \in q(\ell)$ as: $d(i, m) = \delta_{m\ell}$. The target outputs $d(i, \ell)$ only depend on the current input vector $v_i$ and the considered output unit, and not on the classification of the previous one. The difference between criterion (12) and that of a memoryless machine is the additional index $k_n$ which takes account of the previous decision. Collecting all terms depending on the same indexes, (12) can thus be rewritten as:

$$E = \frac{1}{2} \sum_{i=1}^{J} \sum_{k=1}^{K} \sum_{\ell=1}^{K} \sum_{m=1}^{K} n_{ik\ell} \cdot \left[ g(i, k, m) - d(i, m) \right]^2 , \tag{13}$$

where $J = I$ if the *MLP* input is context independent and $J = I^{(2p+1)}$ if a $2p+1$ contextual window is used; $n_{ik\ell}$ represents the number of times $v_i$ has been classified

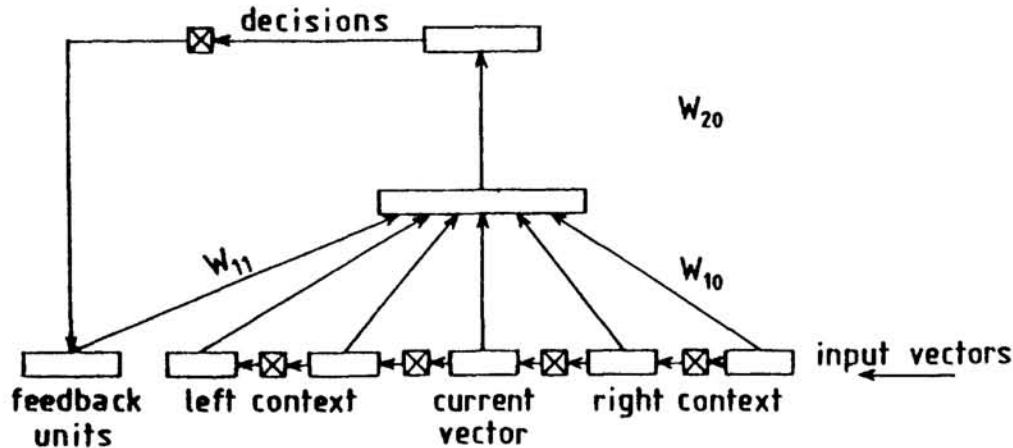

Figure 1: Recurrent and Context-Sensitive MLP ($\boxtimes$ = delay)

in $q(\ell)$ while the previous vector was known to belong to class $q(k)$. Thus, **whatever the MLP topology is**, i.e. the number of its hidden layers and of units per layer, the optimal output values $g_{opt}(i, k, m)$ are obtained by canceling the partial derivative of $E$ versus $g(i, k, m)$. It can easily be proved that, doing so, **the optimal values for the outputs** are then

$$g_{opt}(i, k, m) = \frac{n_{ikm}}{\sum_{\ell=1}^{K} n_{ik\ell}} . \tag{14}$$

The optimal $g(i, k, m)$'s obtained from the minimization of the *MLP* criterion are thus the estimates of the Bayes probabilities, i.e. the discriminant local probabilities defined by (7) if no context is used and by (10) in the contextual case.

It is important to keep in mind that these optimal values can be reached only provided the *MLP* contains enough parameters and does not get stuck into a local minimum during the training.

A convenient way to generate the $g(i, k, \ell)$ is to modify its input as follows. For each $v_{i_n}$, an extended vector $V_{i_n} = (v_{i_n}^+, v_{i_n})$ is formed where $v_{i_n}^+$ is an extra input vector containing the information on the decision taken at time $n - 1$. Since output information is fed back in the input field, such an *MLP* has a *recurrent* topology. The final architecture of the corresponding *MLP* (with *contextual information and output feedback*) is represented in Figure 1 and is similar in design to the net developed in [Jordan, 1986] to produce output pattern sequences.
The main advantage of this topology, when compared with other recurrent models proposed for sequential processing [Elman 1988, Watrous,1987], over and above the possible interpretation in terms of *HMM*, is the control of the information fed back during the training. Indeed, since the training data consists of consecutive labeled speech frames, the correct sequence of output states is known and the training is supervised by providing the correct information.

Replacing in (13) $d(i, m)$ by the optimal values (14) provides a new criterion where the target outputs depend now on the current vector, the considered output and

the classification of the previous vector:

$$E^* = \frac{1}{2} \sum_{i=1}^{J} \sum_{k=1}^{K} \sum_{\ell=1}^{K} \sum_{m=1}^{K} n_{ik\ell} \cdot \left[ g(i,k,m) - \frac{n_{ikm}}{\sum_{\ell=1}^{K} n_{ik\ell}} \right]^2 , \qquad (15)$$

and it is clear (by canceling the partial derivative of $E^*$ versus $g(i,k,m)$) that the lower bound for $E^*$ is reached for the same optimal outputs as (14) but is now equal to zero, what provides a very useful control parameter during the training phase.

It is evident that these results directly follow from the minimized criterion and not from the topology of the model. In that way, it is interesting to note that the same optimal values (14) may also result from other criteria as, for instance, the entropy [Hinton, 1987] or relative entropy [Solla et al., 1988] of the targets with respect to outputs. Indeed, in the case of relative entropy, e.g., criterion (12) is changed in:

$$E_e = \sum_{n=1}^{N} \sum_{\ell=1}^{K} \left[ d(i_n, \ell) . \ln \frac{d(i_n, \ell)}{g(i_n, k_n, \ell)} + (1 - d(i_n, \ell)) . \ln \left( \frac{1 - d(i_n, \ell)}{1 - g(i_n, k_n, \ell)} \right) \right] ,$$
$$(16)$$

and canceling its partial derivative versus $g(i,k,m)$ yields the optimal values (14). In that case, the optimal outputs effectively correspond now to $E_{e,min} = 0$.

Of course, since these results are independent of the topology of the models, they remain also valid for linear discriminant functions but, in that case, it is not guaranteed that the optimal values (14) can be reached. However, it has to be noted that in some particular cases, even for not linearly separable classes, these optimal values are already obtained with linear discriminant functions (and thus with a one layered perceptron trained according to an *LMS* criterion).

It is also important to point out that the same kind of recurrent *MLP* could also be used to estimate local probabilities of higher order Markov models where the local contribution in (9) are no longer assumed dependent on the previous state only but also on several preceding ones. This is easily implemented by extending the input field to the information related to these preceding classifications. Another solution is to represent, in the same extra input vector, a weighted sum (e.g. exponentially decreasing with time) of the preceding outputs [Jordan, 1986].

## CONCLUSION

Discrimination is an essential requirement in speech recognition and is not incorporated in the standard *HMM*. A *discriminant HMM* has been described and links between this new model and a *recurrent MLP* have been shown. Recurrence permits to take account of the sequential information in the output sequence. Moreover, input contextual information is also easily captured by extending the input field. It has finally been proved that the local probabilites of the *discriminant HMM* may be computed (or interpolated) by the particular *MLP* so defined.

# References

[1] Bahl L.R., Brown P.F., de Souza P.V. & Mercer R.L. (1986). Maximum Mutual Information Estimation of Hidden Markov Model Parameters for Speech Recognition, *Proc.ICASSP-86*, Tokyo, pp.49-52,

[2] Bourlard H., Kamp Y., Ney H. & Wellekens C.J. (1985). Speaker-Dependent Connected Speech Recognition via Dynamic Programming and Statistical Methods, *Speech and Speaker Recognition*, Ed. M.R. Schroeder, KARGER,

[3] Bourlard H. & Wellekens C.J. (1989). Speech Pattern Discrimination and Multilayer Perceptrons, *Computer, Speech and Language*, **3**, (to appear),

[4] Brown P. (1987). *The Acoustic-Modeling Problem in Automatic Speech Recognition*, Ph.D. thesis, Comp.Sc.Dep., Carnegie-Mellon University,

[5] Elman J.L. (1988). Finding Structure in Time, *CRL Technical Report 8801, UCSD, Report*,

[6] Hinton G.E. (1987). Connectionist Learning Procedures, Technical Report CMU-CS-87-115,

[7] Jelinek F. (1976). Continuous Recognition by Statistical Methods, *Proceedings IEEE*, vol. 64, no.4, pp. 532-555,

[8] Jordan M.L. (1986). Serial Order: A Parallel Distributed Processing Approach, UCSD, Tech. Report 8604,

[9] Lippmann R.P. (1987). An Introduction to Computing with Neural Nets, *IEEE ASSP Magazine*, vol. 4, pp. 4-22,

[10] Ney H. (1984). The use of a one-stage dynamic programming algorithm for connected word recognition. *IEEE Trans. ASSP* vol. 32, pp.263-271,

[11] Rumelhart D.E., Hinton G.E. & Williams R.J. (1986). Learning Internal Representations by Error Propagation , *Parallel Distributed Processing. Exploration of the Microstructure of Cognition. vol. 1: Foundations*, Ed. D.E.Rumelhart & J.L.McClelland, MIT Press,

[12] Sejnowski T.J. & Rosenberg C.R. (1987). Parallel Networks that Learn to Pronounce English Text. *Complex Systems*, vol. 1, pp. 145-168,

[13] Solla S.A., Levin E. & Fleisher M. (1988). Accelerated Learning in Layered Neural Networks, AT&T Bell Labs. Manuscript,

[14] Waibel A., Hanazawa T., Hinton G., Shikano K. & Lang, K. (1988). Phoneme Recognition Using Time-Delay Neural Networks, *Proc. ICASSP-88*, New York,

[15] Watrous R.L. & Shastri L. (1987). Learning phonetic features using connectionist networks: an experiment in speech recognition, *Proceedings of the First International Conference on Neural Networks*, IV -381-388, San Diego, CA,
